# Learning with Consistency between Inductive Functions and Kernels

**Haixuan Yang**[1,2]   **Irwin King**[1]   **Michael R. Lyu**[1]
[1]Department of Computer Science & Engineering
The Chinese University of Hong Kong
{hxyang,king,lyu}@cse.cuhk.edu.hk

[2]Department of Computer Science
Royal Holloway University of London
haixuan@cs.rhul.ac.hk

## Abstract

Regularized Least Squares (*RLS*) algorithms have the ability to avoid over-fitting problems and to express solutions as kernel expansions. However, we observe that the current *RLS* algorithms cannot provide a satisfactory interpretation even on the penalty of a constant function. Based on the intuition that a good kernel-based inductive function should be consistent with both the data and the kernel, a novel learning scheme is proposed. The advantages of this scheme lie in its corresponding Representer Theorem, its strong interpretation ability about what kind of functions should not be penalized, and its promising accuracy improvements shown in a number of experiments. Furthermore, we provide a detailed technical description about heat kernels, which serves as an example for the readers to apply similar techniques for other kernels. Our work provides a preliminary step in a new direction to explore the varying consistency between inductive functions and kernels under various distributions.

## 1   Introduction

Regularized Least Squares (*RLS*) algorithms have been drawing people's attention since they were proposed due to their ability to avoid over-fitting problems and to express solutions as kernel expansions in terms of the training data [4, 9, 12, 13]. Various modifications of *RLS* are made to improve its performance either from the viewpoint of manifold [1] or in a more generalized form [7, 11]. However, despite these modifications, problems still remain. We observe that the previous *RLS*-related work has the following problem:

**Over Penalization.** For a constant function $f = c$, a nonzero term $||f||_K$ is penalized in both *RLS* and *LapRLS* [1]. As a result, for a distribution generalized by a nonzero constant function, the resulting regression function by both *RLS* and *LapRLS* is not a constant as illustrated in the left diagram in Fig. 1. For such situations, there is an over-penalization.

In this work, we aim to provide a new viewpoint for supervised or semi-supervised learning problems. By such a viewpoint we can provide a general condition under which constant functions should not be penalized. The basic idea is that, if a learning algorithm can learn an inductive function $f(x)$ from examples generated by a joint probability distribution P on $X \times \mathbb{R}$, then the learned function $f(x)$ and the marginal $P_X$ represents a new distribution on $X \times \mathbb{R}$, from which there is a re-learned function $r(x)$. The re-learned function should be consistent with the learned function in the sense that the expected difference on distribution $P_X$ is small. Because the re-learned function depends on the underlying kernel, the difference $f(x) - r(x)$ depends on $f(x)$ and the kernel, and from this point of view, we name this work.

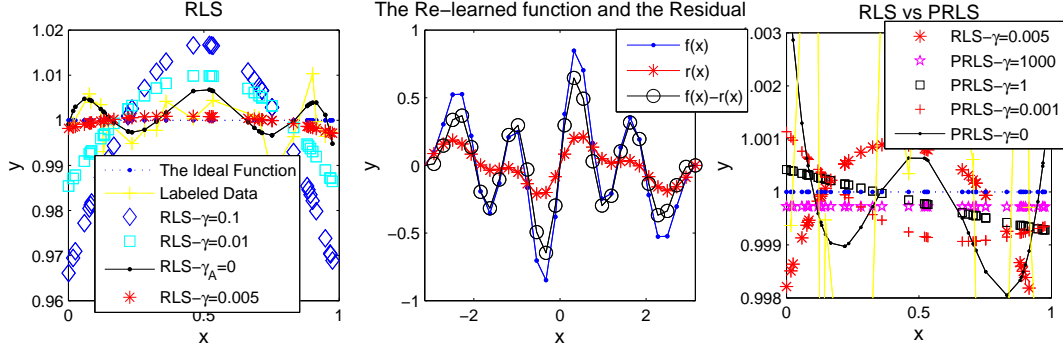

Figure 1: Illustration for over penalization. Left diagram: The training set contains 20 points, whose $x$ is randomly drawn from the interval $[0\ 1]$, whereas the test set contains another 20 points, and $y$ is generated by $1 + 0.005\varepsilon$, $\varepsilon \sim N(0,1)$. The over penalized constant functions in the term $||f||_K$ cause the phenomena that smaller $\gamma$ can achieve better results. On the other hand, the over-fitting phenomenon when $\gamma = 0$ suggests the necessity of the regularization term. Based on these observations, an appropriate penalization on a function is expected. Middle diagram: $r(x)$ is very smooth, and $f(x) - r(x)$ remains the uneven part of $f(x)$; therefore $f(x) - r(x)$ should be penalized while $f$ is over penalized in $||f||_K$. Right diagram: the proposed model has a stable property so that a large variant of $\gamma$ results in small changes of the curves, suggesting a right way of penalizing functions.

## 2 Background

**The RKHS Theory** enables us to express solutions of *RLS* as kernel expansions in terms of the training data. Here we give a brief description of the concepts. For a complete discussion, see [2]. Let $X$ be a compact domain or manifold, $\nu$ be a Borel measure on $X$, and $K : X \times X \to \mathbb{R}$ be a Mercer kernel, then there is an associated Hilbert space RKHS $\mathcal{H}_K$ of functions $X \to \mathbb{R}$ with the corresponding norm $|| \cdot ||_K$. $\mathcal{H}_K$ satisfies the *reproducing property*, i.e., for all $f \in \mathcal{H}_K$, $f(x) = \langle K_x, f \rangle$, where $K_x$ is the function $K(x, \cdot)$. Moreover, an operator $L_K$ can be defined on $\mathcal{H}_K$ as: $(L_K f)(x) = \int_X f(y) K(x, y) d\nu(y)$, where $\mathcal{L}_\nu^2(X)$ is the Hilbert space of square integrable functions on $X$ with the scalar product $\langle f, g \rangle_\nu = \int_X f(x) g(x) d\nu(x)$.

Given a Mercer kernel and a set of labeled examples $(x_i, y_i)$ $(i = 1, ..., l)$, there are two popular inductive learning algorithms: *RLS* [12, 13] and the Nadaraya-Watson Formula [5, 8, 14]. By the standard Tikhonov regularization, *RLS* is a special case of the following functional extreme problem:

$$f^* = \arg \min_{f \in \mathcal{H}_K} \frac{1}{l} \sum_{i=1}^{l} V(x_i, y_i, f) + \gamma ||f||_K^2 \qquad (1)$$

where $V$ is some loss function.

**The Classical Representer Theorem** states that the solution to this minimization problem exists in $\mathcal{H}_K$ and can be written as

$$f^*(x) = \sum_{i=1}^{l} \alpha_i K(x_i, x). \qquad (2)$$

Such a Representer Theorem is general because it plays an important role in both *RLS* in the case when $V(x, y, f) = (y - f(x))^2$, and SVM in the case when $V(x, y, f) = \max(0, 1 - yf(x))$.

**The Nadaraya-Watson Formula** is based on local weighted averaging, and it comes with a closed form:

$$r(x) = \sum_{i=1}^{l} y_i K(x, x_i) / \sum_{i=1}^{l} K(x, x_i). \tag{3}$$

The formula has a similar appearance as Eq. (2), but it plays an important role in this paper because we can write it in an integral form which makes our idea technically feasible as follows. Let $p(x)$ be a probability density function over $X$, $P(x)$ be the corresponding cumulative distribution function, and $f(x)$ be an inductive function. We observe that, if $(x_i, f(x_i))(i = 1, 2, \ldots, l)$ are sampled from the function $y = f(x)$, then

**A Re-learned Function** can be expressed as

$$r(x) = \lim_{l \to \infty} \frac{\sum_{i=1}^{l} f(x_i) K(x, x_i)}{\sum_{i=1}^{l} K(x, x_i)} = \frac{\int_X f(\alpha) K(x, \alpha) dP(\alpha)}{\int_X K(x, \alpha) dP(\alpha)} = \frac{L_K(f)}{\int_X K(x, \alpha) dP(\alpha)}, \tag{4}$$

based on $f(x)$ and $P(x)$. From this form, we show two points: (1) If $r(x) = f(x)$, then $f(x)$ is completely predicted by itself through the Nadaraya-Watson Formula, and so $f(x)$ is considered to be completely consistent with the kernel $K(x, y)$; if $r(x) \neq f(x)$, then the difference $||f(x) - r(x)||_K$ can measure how badly $f(x)$ is consistent with the kernel $K(x, y)$ and (2) Intuitively $r(x)$ can also be understood as the smoothed function of $f(x)$ through a kernel $K$. Consequently, $f(x) - r(x)$ represents the intrinsically uneven part of $f(x)$, which we will penalize. This intuition is illustrated in the middle diagram in Fig. 1.

Throughout this paper, we assume that $\int_X K(x, \alpha) dP(\alpha)$ is a constant, and for simplicity all kernels are normalized by $K / \int_X K(x, \alpha) dP(\alpha)$ so that $r(x) = L_K(f)$. Moreover, we assume that $X$ is compact, and the measure $\nu$ is specified as $P(x)$.

## 3 Partially-penalized Regularization

For a given kernel $K$ and an inductive function $f$, $L_K(f)$ is the prediction function produced by $K$ through the Nadaraya-Watson Formula. Based on Eq. (1), penalizing the inconsistent part $f(x) - L_K(f)$ leads to the following Partially-penalized Regularization problem:

$$f^* = \arg \min_{f \in \mathcal{H}_K} \frac{1}{l} \sum_{i=1}^{l} V(x_i, y_i, f) + \gamma ||f - L_K(f)||_K^2. \tag{5}$$

To obtain a Representer Theorem, we need one assumption.

**Assumption 1** *Let $f_1, f_2 \in \mathcal{H}_K$. If $\langle f_1, f_2 \rangle_K = 0$, then $||f_1 - L_K(f_1) + f_2 - L_K(f_2)||_K^2 = ||f_1 - L_K(f_1)||_K^2 + ||f_2 - L_K(f_2)||_K^2$.*

It is well-known that the operator $L_K$ is compact, self-adjoint, and positive with respect to $\mathcal{L}_\nu^2(X)$, and by the Spectral Theorem [2, 3], its eigenfunctions $e_1(x), e_2(x), \ldots$ form an orthogonal basis of $\mathcal{L}_\nu^2(X)$ and the corresponding eigenvalues $\lambda_1 \geq \lambda_2, \ldots$ are either finitely many that are nonzero, or there are infinitely many, in which case $\lambda_k \to 0$. Let $f_1 = \sum_i a_i e_i(x)$, $f_2 = \sum_i b_i e_i(x)$, then $f_1 - L_K(f_1) = \sum_i a_i e_i(x) - L_K(\sum_i a_i e_i(x)) = \sum_i a_i e_i(x) - \sum_i \lambda_i a_i e_i(x) = \sum_i (1 - \lambda_i) a_i e_i(x)$, and similarly, $f_2 - L_K(f_2) = \sum_i (1 - \lambda_i) b_i e_i(x)$. By the discussions in [1], we have $\langle e_i, e_j \rangle_\nu = 0$ if $i \neq j$, and $\langle e_i, e_i \rangle_\nu = 1$; $\langle e_i, e_j \rangle_K = 0$ if $i \neq j$, and $\langle e_i, e_i \rangle_K = \frac{1}{\lambda_i}$. If we consider the situation that $a_i, b_i \geq 0$ for all $i \geq 1$, then $\langle f_1, f_2 \rangle_K = 0$ implies that $a_i b_i = 0$ for all $i \geq 1$, and consequently $\langle f_1 - L_K(f_1), f_2 - L_K(f_2) \rangle_K = \sum_i (1 - \lambda_i)^2 a_i b_i \langle e_i(x), e_i(x) \rangle_K = 0$. Therefore, under some constrains, this assumption is a fact. Under this assumption, we have a Representer Theorem.

**Theorem 2** *Let $\mu_j(x)$ be a basis in $\mathcal{H}_0$ of the operator $I - L_K$, i.e., $\mathcal{H}_0 = \{f \in \mathcal{H}_K | f - L_K(f) = 0\}$. Under Assumption 1, the minimizer of the optimization problem in Eq. (5) is*

$$f^*(x) = \sum_{j=1}^{o} \beta_j \mu_j(x) + \sum_{i=1}^{l} \alpha_i K(x_i, x) \tag{6}$$

**Proof of the Representer Theorem.** Any function $f \in \mathcal{H}_K$ can be uniquely decomposed into a component $f_{||}$ in the linear subspace spanned by the kernel functions $\{K(x_i, \cdot)\}_{i=1}^{l}$, and a component $f_{\perp}$ orthogonal to it. Thus, $f = f_{||} + f_{\perp} = \sum_{i=1}^{l} \alpha_i K(x_i, \cdot) + f_{\perp}$. By the reproducing property and the fact that $\langle f_{\perp}, K(x_i, \cdot)\rangle = 0$ for $1 \le i \le l$, we have

$$f(x_j) = \langle f, K(x_j, \cdot)\rangle = \langle \sum_{i=1}^{l} \alpha_i K(x_i, \cdot), K(x_j, \cdot)\rangle + \langle f_{\perp}, K(x_j, \cdot)\rangle = \langle \sum_{i=1}^{l} \alpha_i K(x_i, \cdot), K(x_j, \cdot)\rangle.$$

Thus the empirical terms involving the loss function in Eq. (5) depend only on the value of the coefficients $\{\alpha_i\}_{i=1}^{l}$ and the gram matrix of the kernel function. By Assumption 1, we have

$$
\begin{aligned}
||f - L_K(f)||_K^2 &= \quad || \sum_{i=1}^{l} \alpha_i K(x_i, \cdot) - L_K(\sum_{i=1}^{l} \alpha_i K(x_i, \cdot))||_K^2 + ||f_{\perp} - L_K(f_{\perp})||_K^2 \\
&\ge \quad || \sum_{i=1}^{l} \alpha_i K(x_i, \cdot) - L_K(\sum_{i=1}^{l} \alpha_i K(x_i, \cdot))||_K^2.
\end{aligned}
$$

It follows that the minimizer of Eq. (5) must have $||f_{\perp} - L_K(f_{\perp})||_K^2 = 0$, and therefore admits a representation $f^*(x) = f_{\perp} + \sum_{i=1}^{l} \alpha_i K(x_i, x) = \sum_{j=1}^{o} \beta_j \mu_j(x) + \sum_{i=1}^{l} \alpha_i K(x_i, x)$.

## 3.1 Partially-penalized Regularized Least Squares (*PRLS*) Algorithm

In this section, we focus our attention in the case that $V(x_i, y_i, f) = (y_i - f(x_i))^2$, i.e, the Regularized Least Squares algorithm. In our setting, we aim to solve:

$$\min_{f \in \mathcal{H}_K} \frac{1}{l} \sum (y_i - f(x_i))^2 + \gamma ||f - L_K(f)||_K^2. \tag{7}$$

By the Representer Theorem, the solution to Eq. (7) is of the following form:

$$f^*(x) = \sum_{j=1}^{o} \beta_j \mu_j(x) + \sum_{i=1}^{l} \alpha_i K(x_i, x). \tag{8}$$

By the proof of Theorem 2, we have $f_{\perp} = \sum_{j=1}^{o} \beta_j \mu_j(x)$ and $\langle f_{\perp}, \sum_{i=1}^{l} \alpha_i K(x_i, x)\rangle_K = 0$. By Assumption 1 and the fact that $f_{\perp}$ belongs to the null space $\mathcal{H}_0$ of the operator $I - L_K$, we have

$$
\begin{aligned}
||f^* - L_K(f^*)||_K^2 &= ||f_{\perp} - L_K(f_{\perp})||_K^2 + || \sum_{i=1}^{l} \alpha_i K(x_i, x) - L_K(\sum_{i=1}^{l} \alpha_i K(x_i, x))||_K^2 \\
&= || \sum_{i=1}^{l} \alpha_i K(x_i, x) - \sum_{i=1}^{l} \alpha_i L_K(K(x_i, x))||_K^2 = \alpha^T (K - 2K' + K'')\alpha,
\end{aligned} \tag{9}
$$

where $\alpha = [\alpha_1, \alpha_2, \ldots, \alpha_l]^T$, $K$ is the $l \times l$ gram matrix $K_{ij} = K(x_i, x_j)$, $K'$ and $K''$ are reconstructed $l \times l$ matrices $K'_{ij} = \langle K(x_i, x), L_K(K(x_j, x))\rangle_K$, and $K''_{ij} = \langle L_K(K(x_i, x)), L_K(K(x_j, x))\rangle_K$. Substituting Eq. (8) and Eq. (9) to the problem in Eq. (7), we arrive at the following quadratic objective function of the $l$-dimensional variable $\alpha$ and $o$-dimensional variable $\beta = [\beta_1, \beta_2, \ldots, \beta_o]^T$:

$$[\alpha^*, \beta^*] = \arg\min \frac{1}{l}(Y - K\alpha - \Psi\beta)^T (Y - K\alpha - \Psi\beta) + \gamma \alpha^T (K - 2K' + K'')\alpha, \tag{10}$$

where $\Psi$ is an $l \times o$ matrix $\Psi_{ij} = \mu_j(x_i)$, and $Y = [y_1, y_2, \ldots, y_l]^T$. Taking derivatives with respect to $\alpha$ and $\beta$, since the derivative of the objective function vanishes at the minimizer, we obtain

$$(\gamma l(K - 2K' + K'') + K^2)\alpha + K\Psi\beta = KY, \quad \Psi^T(Y - K\alpha - \Psi\beta) = 0. \tag{11}$$

In the term $||f - L_K(f)||$, $f$ is subtracted by $L_K(f)$, and so it partially penalized. For this reason, the resulting algorithm is referred as Partially-penalized Regularized Least Squares algorithm (*PRLS*).

## 3.2  The *PLapRLS* Algorithm

The idea in the previous section can also be extended to *LapRLS* in the manifold regularization framework [1]. In the manifold setting, the smoothness on the data adjacency graph should be considered, and Eq. (5) is modified as

$$f^* = \arg\min_{f \in \mathcal{H}_K} \frac{1}{l} \sum_{i=1}^{l} V(x_i, y_i, f) + \gamma_A \|f - L_K(f)\|_K^2 + \frac{\gamma_I}{(u+l)^2} \sum_{i,j=1}^{l+u} (f(x_i) - f(x_j))^2 W_{ij}, \quad (12)$$

where $W_{ij}$ are edge weights in the data adjacency. From $W$, the graph Laplacian $L$ is given by $L = D - W$, where $D$ is the diagonal matrix with $D_{ii} = \sum_{j=1}^{l+u} W_{ij}$. For this optimization problem, the result in Theorem 2 can be modified slightly as:

**Theorem 3** *Under Assumption 1, the minimizer of the optimization problem in Eq. (12) admits an expansion*

$$f^*(x) = \sum_{j=1}^{o} \beta_j \mu_j(x) + \sum_{i=1}^{l+u} \alpha_i K(x_i, x). \quad (13)$$

Following Eq. (13), we continue to optimize the $(l + u)$-dimensional variable $\alpha = [\alpha_1, \alpha_2, \ldots, \alpha_{l+u}]\alpha$ and the $o$-dimensional variable $\beta = [\beta_1, \beta_2, \ldots, \beta_o]^T$. In a similar way as the previous section and *LapRLS* in [1], $\alpha$ and $\beta$ are determined by the following linear systems:

$$\begin{cases} (KJK + \lambda_1(K - 2K' + K'') + \lambda_2 KLK)\alpha + (KJ\Psi + \lambda_2 KL\Psi)\beta = KJY, \\ (\Psi'JK - \lambda_2\Psi'LK)\alpha + (\Psi'\Psi - \lambda_2\Psi'L\Psi)\beta = \Psi' * Y, \end{cases} \quad (14)$$

where $K$, $K'$, $K''$ are the $(l+u) \times (l+u)$ Gram matrices over labeled and unlabeled points; $Y$ is an $(l + u)$ dimensional label vector given by: $Y = [y_1, y_2, \ldots, y_l, 0, \ldots, 0]$, $J$ is an $(l + u) \times (l + u)$ diagonal matrix given by $J = \text{diag}(1, 1, \ldots, 1, 0, \ldots, 0)$ with the first $l$ diagonal entries as 1 and the rest 0, and $\Psi$ is an $(l + u) \times o$ matrix $\Psi_{ij} = \mu_j(x_i)$.

# 4  Discussions

## 4.1  Heat Kernels and the Computation of $K'$ and $K''$

In this section we will illustrate the computation of $K'$ and $K''$ in the case of heat kernels. The basic facts about heat kernels are excerpted from [6], and for more materials, see [10].

Given a manifold $\mathcal{M}$ and points $x$ and $y$, the heat kernel $K_t(x, y)$ is a special solution to the heat equation with a special initial condition called the delta function $\delta(x-y)$. More specifically, $\delta(x-y)$ describes a unit heat source at position $y$ with no heat in other positions. Namely, $\delta(x - y) = 0$ for $x \neq y$ and $\int_{-\infty}^{+\infty} \delta(x - y)dx = 1$. If we let $f_0(x, 0) = \delta(x - y)$, then $K_t(x, y)$ is a solution to the following differential equation on a manifold $\mathcal{M}$:

$$\frac{\partial f}{\partial t} - \mathcal{L}f = 0, \quad f(x, 0) = f_0(x), \quad (15)$$

where $f(x, t)$ is the temperature at location $x$ at time $t$, beginning with an initial distribution $f_0(x)$ at time zero, and $\mathcal{L}$ is the *Laplace-Beltrami operator*. Equation (15) describes the heat flow throughout a geometric manifold with initial conditions.

**Theorem 4** *Let $\mathcal{M}$ be a complete Riemannian manifold. Then there exists a function $K \in C^\infty(\mathbb{R}_+ \times \mathcal{M} \times \mathcal{M})$, called the heat kernel, which satisfies the following properties for all $x, y \in \mathcal{M}$, with $K_t(x, y) = K(t, x, y)$: (1) $K_t(x, y)$ defines a Mercer kernel. (2) $K_t(x, y) = \int_{\mathcal{M}} K_{t-s}(x, z)K_s(z, y)dz$ for any $s > 0$. (3) The solution to Eq. (15) is $f(x, t) = \int_{\mathcal{M}} K_t(x, y)f_0(y)dy$. (4) $1 = \int_{\mathcal{M}} K_t(x, y)1dy$ and (5) When $\mathcal{M} = \mathbb{R}^m$, $\mathcal{L}f$ is simplified as $\sum_i \frac{\partial^2 f}{\partial x_i^2}$, and the heat kernel takes the Gaussian RBF form $K_t(x, y) = (4\pi t)^{-\frac{m}{2}} e^{-\frac{\|x-y\|^2}{4t}}$.*

$K'$ and $K''$ can be computed as follows:

$$
\begin{aligned}
K'_{ij} &= \langle K_t(x_i, x), L_K(K_t(x_j, x)) \rangle_K \quad \text{(by definition)} \\
&= L_K(K_t(x_j, x))|_{x=x_i} \quad \text{(by the reproducing property of a Mercer kernel)} \\
&= \int_X K_t(x_j, y) K_t(x_i, y) d\nu(y) \quad \text{(by the definition of } L_K) \\
&= K_{2t}(x_i, x_j) \quad \text{(by Property 2 in Theorem 4)}
\end{aligned}
\tag{16}
$$

Based on the fact that $L_K$ is self-adjoint, we can similarly derive $K''_{ij} = K_{3t}(x_i, x_j)$. For other kernels, $K'$ and $K''$ can also be computed.

## 4.2 What should not be penalized?

From Theorem 2, we know that the functions in the null space $\mathcal{H}_0 = \{ f \in \mathcal{H}_K | f - L_K(f) = 0 \}$ should not be penalized. Although there may be looser assumptions that can guarantee the validity of the result in Theorem 2, there are two assumptions in this work: $X$ is compact and $\int_X K(x, \alpha) dP(\alpha)$ in Eq. (4) is a constant. Next we discuss the constant functions and the linear functions.

**Should constant functions be penalized?** Under the two assumptions, a constant function $c$ should not be penalized, because $c = \int_X cK(x, \alpha)p(\alpha)d\alpha / \int_X K(x, \alpha)p(\alpha)d\alpha$, i.e., $c \in \mathcal{H}_0$. For heat kernels, if $P(x)$ is uniformly distributed on $\mathcal{M}$, then by Property 4 in Theorem 4, $\int_X K(x, \alpha) dP(\alpha)$ is a constant, and so $c$ should not be penalized.

For polynomial kernels, the theory cannot guarantee that constant functions should not be penalized even with a uniform distribution $P(x)$. For example, considering the polynomial kernel $xy + 1$ in the interval $X = [0\ 1]$ and the uniform distribution on $X$, $\int_X (xy+1) dP(y) = \int_0^1 (xy+1) dy = x/2 + 1$ is not a constant. As a counter example, we will show in Section 5.3 that not penalizing constant functions in polynomial kernels will result in much worse accuracy. The reason for this phenomenon is that constant functions may not be smooth in the feature space produced by the polynomial kernel under some distributions. The readers can deduce an example for $p(x)$ such that $\int_0^1 (xy + 1) dP(y)$ happens to be a constant.

**Should linear function $a^T x$ be penalized?** In the case when $X$ is a closed ball $B_r$ with radius $r$ when $P(x)$ is uniformly distributed over $B_r$ and when $K$ is the Gaussian RBF kernel, then $a^T x$ should not be penalized when $r$ is big enough. [1] Since $r$ is big enough, we have $\int_{\mathbb{R}^n} \cdot dx \approx \int_{B_r} \cdot dx$ and $\int_{B_r} K_t(x, y) dy \approx 1$, and so $a^T x = \int_{\mathbb{R}^n} K_t(x, y) a^T y dy \approx \int_{B_r} K_t(x, y) a^T y dy \approx L_K(a^T x)$. Consequently $||a^T x - L_K(a^T x)||_K$ will be small enough, and so the linear function $a^T x$ needs not be penalized. For other kernels, other spaces, or other $P_X$, the conclusion may not be true.

# 5 Experiments

In this section, we evaluate the proposed algorithms *PRLS* and *PLapRLS* on a toy dataset (size: 40), a medium-sized dataset (size: 3,119), and a large-sized dataset (size: 20,000), and provide a counter example for constant functions on another dataset (size: 9,298). We use the Gaussian RBF kernels in the first three datasets, and use polynomial kernels to provide a counter example on the last dataset. Without any prior knowledge about the data distribution, we assume that the examples are uniformly distributed, and so constant functions are considered to be in $\mathcal{H}_0$ for the Gaussian RBF kernel, but linear functions are not considered to be in $\mathcal{H}_0$ since it is rare for data to be distributed uniformly on a large ball. The data and results for the toy dataset are illustrated in the left diagram and the right diagram in Fig. 1.

## 5.1 UCI Dataset Isolet about Spoken Letter Recognition

We follow the same semi-supervised settings as that in [1] to compare *RLS* with *PRLS*, and compare *LapRLS* with *PLapRLS* on the Isolet database. The dataset contains utterances of 150 subjects who

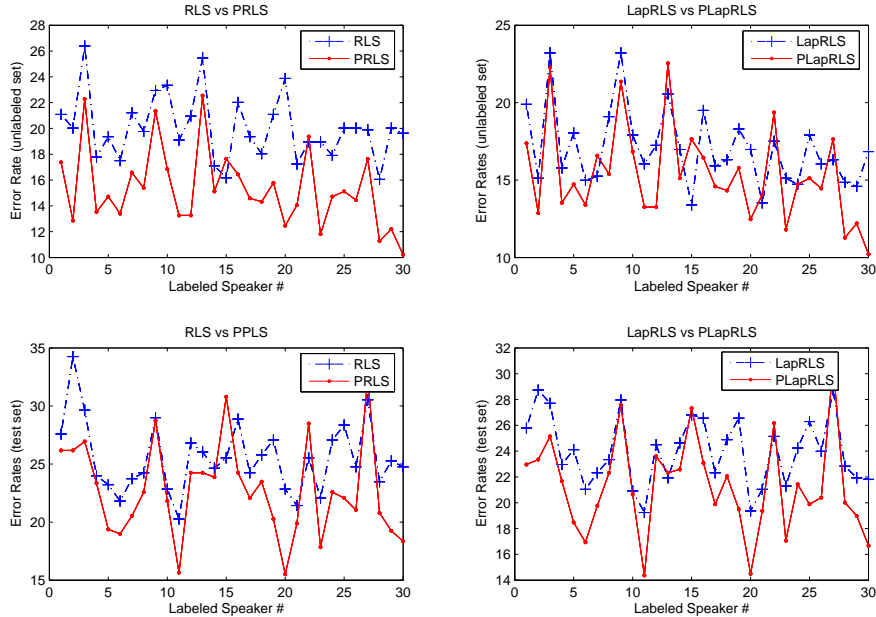

Figure 2: Isolet Experiment

pronounced the name of each letter of the English alphabet twice. The speakers were grouped into 5 sets of 30 speakers each. The data of the first 30 speakers forms a training set of 1,560 examples, and that of the last 29 speakers forms the test set. The task is to distinguish the first 13 letters from the last 13. To simulate a real-world situation, 30 binary classification problems corresponding to 30 splits of the training data where all 52 utterances of one speaker were labeled and all the rest were left unlabeled. All the algorithms use Gaussian RBF kernels. For *RLS* and *LapRLS*, the results were obtained with width $\sigma = 10$, $\gamma l = 0.05$, $\gamma_A l = \gamma_I l/(u+l)^2 = 0.005$. For *PRLS* and *PLapRLS*, the results were obtained with width $\sigma = 4$, $\gamma l = 0.01$, and $\gamma_A l = \gamma_I l/(u+l)^2 = 0.01$. In Fig. 2, we can see that both *PRLS* and *PLapRLS* make significant performance improvements over their corresponding counterparts on both unlabeled data and test set.

## 5.2 UCI Dataset Letter about Printed Letter Recognition

In Dataset Letter, there are 16 features for each example, and there are 26 classes representing the upper case printed letters. The first 400 examples were taken to form the training set. The remaining 19,600 examples form the test set. The parameters are set as follows: $\sigma = 1, \gamma l = \gamma_A(l+u) = 0.25$, and $\gamma_I l/(u+l)^2 = 0.05$. For each of the four algorithms *RLS*, *PRLS*, *LapRLS*, and *PLapRLS*, for each of the 26 one-versus-all binary classification tasks, and for each of 10 runs, two examples for each class were randomly labeled. For each algorithm, the averages over all the 260 one-versus-all binary classification error rates for unlabeled 398 examples and test set are listed respectively as follows: (5.79%, 5.23%) for *RLS*, (5.12%, 4.77%) for *PRLS*, (0%, 2.96%) for *LapRLS*, and (0%, 3.15%) for *PLapRLS* respectively. From the results, we can see that *RLS* is improved on both unlabeled examples and test set. The fact that there is no error in the total 260 tasks for *LapRLS* and *PLapRLS* on unlabeled examples suggests that the data is distributed in a curved manifold. On a curved manifold, the heat kernels do not take the Gaussian RBF form, and so *PLapRLS* using the Gaussian RBF form cannot achieve its best. This is the reason why we can observe that *PLapRLS* is slightly worse than *LapRLS* on the test set. This suggests the need for a vast of investigations on heat kernels on a manifold.

## 5.3 A Counter Example in Handwritten Digit Recognition

Note that, polynomial kernels with degree 3 were used on USPS dataset in [1], and 2 images for each class were randomly labeled. We follow the same experimental setting as that in [1]. For *RLS*, if we

use Eq. (2), then the averages of 45 pairwise binary classification error rates are 8.83% and 8.41% for unlabeled 398 images and 8,898 images in the test set respectively. If constant functions are not penalized, then we should use $f^*(x) = \sum_{i=1}^{l} \alpha_i K(x_i, x) + a$, and the corresponding error rates are 9.75% and 9.09% respectively. By this example, we show that leaving constant functions outside the regularization term is dangerous; however, it is fortunate that we have a theory to guide this in Section 4: if $X$ is compact and $\int_X K(x, \alpha) dP(\alpha)$ in Eq. (4) is a constant, then constant functions should not be penalized.

## 6 Conclusion

A novel learning scheme is proposed based on a new viewpoint of penalizing the inconsistent part between inductive functions and kernels. In theoretical aspects, we have three important claims: (1) On a compact domain or manifold, if the denominator in Eq. (4) is a constant, then there is a new Representer Theorem; (2) The same conditions become a sufficient condition under which constant functions should not be penalized; and (3) under the same conditions, a function belongs to the null space if and only if the function should not be penalized. Empirically, we claim that the novel learning scheme can achieve accuracy improvement in practical applications.

**Acknowledgments**

The work described in this paper was supported by two grants from the Research Grants Council of the Hong Kong Special Administrative Region, China (Project No. CUHK4150/07E) and Project No. CUHK4235/04E). The first author would like to thank Hao Ma for his helpful suggestions, thank Kun Zhang and Wenye Li for useful discussions, and thank Alberto Paccanaro for his support.

## Footnotes

[1]Note that a subset of $\mathbb{R}^n$ is compact if and only if it is closed and bounded. Since $\mathbb{R}^n$ is not bounded, it is not compact, and so the Representer Theorem cannot be established. This is the reason why we cannot talk about $\mathbb{R}^n$ directly.

## References

[1] GMikhail Belkin, Partha Niyogi, and Vikas Sindhwani. Manifold regularization: A geometric framework for learning from labeled and unlabeled examples. *Journal of Machine Learning Research*, 7:2399–2434, 2006.

[2] F. Cucker and S. Smale. On the mathematical foundations of learning. *Bulletin (New Series) of the American Mathematical Society*, 39(1):1–49, 2002.

[3] Lokenath Debnath and Piotr Mikusinski. *Introduction to Hilbert Spaces with Applications*. Academic Press, San Diego, second edition, 1999.

[4] T. Evgeniou, M. Pontil, and T. Poggio. Regularization networks and support vector machines. *Advances in Computational Mathematics*, 13:1–50, 2000.

[5] T. Hastie and C. Loader. Local regression: Automatic kernel carpentry. *Statistical Science*, 8(1):120–129, 1993.

[6] John Lafferty and Guy Lebanon. Diffusion kernels on statistical manifolds. *Journal of Machine Learning Research*, 6:129–163, 2005.

[7] Wenye Li, Kin-Hong Lee, and Kwong-Sak Leung. Generalized regularized least-squares learning with predefined features in a Hilbert space. In *NIPS*, 2006.

[8] E. A. Nadaraya. On estimating regression. *Theory of Probability and Its Applications*, 9(1):141–142, 1964.

[9] R.M. Rifkin and R.A. Lippert. Notes on regularized least-squares. Technical Report 2007-019, Massachusetts Institute of Technology, 2007.

[10] S. Rosenberg. *The Laplacian on a Riemmannian Manifold*. Cambridge University Press, 1997.

[11] Bernhard Schölkopf, Ralf Herbrich, and Alex J. Smola. A generalized representer theorem. In *COLT*, 2001.

[12] I. Schönberg. Spline functions and the problem of graduation. *Proc. Nat. Acad. Sci. USA*, 52:947–950, 1964.

[13] A. N. Tikhonov and V. Y. Arsenin. *Solutions of Ill-posed Problems*. W. H. Winston, 1977.

[14] G. S. Watson. Smooth regression analysis. *Sankhyá, Series A*, 26:359–372, 1964.
